# Weak Learners and Improved Rates of Convergence in Boosting

**Shie Mannor and Ron Meir**
Department of Electrical Engineering
Technion, Haifa 32000, Israel
{shie,rmeir}@{techunix,ee}.technion.ac.il

## Abstract

The problem of constructing weak classifiers for boosting algorithms is studied. We present an algorithm that produces a linear classifier that is guaranteed to achieve an error better than random guessing for any distribution on the data. While this weak learner is not useful for learning in general, we show that under reasonable conditions on the distribution it yields an effective weak learner for one-dimensional problems. Preliminary simulations suggest that similar behavior can be expected in higher dimensions, a result which is corroborated by some recent theoretical bounds. Additionally, we provide improved convergence rate bounds for the generalization error in situations where the empirical error can be made small, which is exactly the situation that occurs if weak learners with guaranteed performance that is better than random guessing can be established.

## 1 Introduction

The recently introduced boosting approach to classification (e.g., [10]) has been shown to be a highly effective procedure for constructing complex classifiers. Boosting type algorithms have recently been shown [9] to be strongly related to other incremental greedy algorithms (e.g., [6]). Although a great deal of numerical evidence suggests that boosting works very well across a wide spectrum of tasks, it is not a panacea for solving classification problems. In fact, many versions of boosting algorithms currently exist (e.g., [4],[9]), each possessing advantages and disadvantages in terms of classification accuracy, interpretability and ease of implementation.

The field of boosting provides two major theoretical results. First, it is shown that in certain situations the *training* error of the classifier formed converges to zero (see (2)). Moreover, under certain conditions, a positive margin can be guaranteed. Second, bounds are provided for the generalization error of the classifier (see (1)). The main contribution of this paper is twofold. First, we present a simple and efficient algorithm which is shown, for *every* distribution on the data, to yield a *linear* classifier with *guaranteed* error which is smaller than $1/2 - \gamma$ where $\gamma$ is strictly positive. This establishes that a weak linear classifier exists. From the theory of boosting [10] it is known that such a condition suffices to guarantee that the training error converges to zero as the number of boosting iterations increases.

In fact, the empirical error with a finite margin is shown to converge to zero if $\gamma$ is sufficiently large. However, the existence of a weak learner with error $1/2 - \gamma$ is not always useful in terms of generalization error, since it applies even to the extreme case where the binary labels are drawn independently at random with equal probability at each point, in which case we cannot expect any generalization. It is then clear that in order to construct useful weak learners, some assumptions need to be made about the data. In this work we show that under certain natural conditions, a useful weak learner can be constructed for one-dimensional problems, in which case the linear hyper-plane degenerates to a point. We speculate that similar results hold for higher dimensional problems, and present some supporting numerical evidence for this. In fact, some very recent results [7] show that this expectation is indeed borne out. The second contribution of our work consists of establishing faster convergence rates for the generalized error bounds introduced recently by Mason *et al.* [8]. These improved bounds show that faster convergence can be achieved if we allow for convergence to a slightly larger value than in previous bounds. Given the guaranteed convergence of the empirical loss to zero (in the limited situations in which we have proved such a bound), such a result may yield a better trade-off between the terms appearing in the bound, offering a better model selection criterion (see Chapter 15 in [1]).

## 2 Construction of a Linear Weak Learner

We recall the basic generalization bound for convex combinations of classifiers. Let $H$ be a class of binary classifiers of VC-dimension $d_v$, and denote by $\mathrm{co}(H)$ the convex hull of $H$. Given a sample $S = \{(x_1, y_1), \ldots, (x_m, y_m)\} \in (\mathcal{X} \times \{-1, +1\})^m$ of $m$ examples drawn independently at random from a probability distribution $D$ over $\mathcal{X} \times \{-1, +1\}$, Schapire *et al.* [10] show that with probability at least $1 - \delta$, for every $f \in \mathrm{co}(H)$ and every $\theta > 0$,

$$\mathbf{P}_D[Yf(X) \leq 0] \leq \mathbf{P}_S[Yf(X) \leq \theta] + O\left(\frac{1}{\sqrt{m}} \left(\frac{d_v \log^2(m/d_v)}{\theta^2} + \log(1/\delta)\right)^{1/2}\right),$$

(1)

where the margin-error $\mathbf{P}_S[Yf(X) \leq \theta]$ denotes the fraction of training points for which $y_i f(x_i) \leq \theta$. Clearly, if the first term can be made small for a large value of the margin $\theta$, a tight bound can be established. Schapire *et al.* [10] also show that if each weak classifier can achieve an error smaller than $1/2 - \gamma$, then

$$\mathbf{P}_S[Yf(X) \leq \theta] \leq \left((1 - 2\gamma)^{1-\theta}(1 + 2\gamma)^{1+\theta}\right)^{T/2},$$

(2)

where $T$ is the number of boosting iterations. Note that if $\gamma > \theta$, the bound decreases to zero exponentially fast. It is thus clear that a large value of $\gamma$ is needed in order to guarantee a small value for the margin-error. However, if $\gamma$ (and thus $\theta$) behaves like $m^{-\beta}$ for some $\beta > 0$, the rate of convergence in the second term in (1) will deteriorate, leading to worse bounds than those available by using standard VC results [11]. What is needed is a characterization of conditions under which the achievable $\theta$ does not decrease rapidly with $m$. In this section we present such conditions for one-dimensional problems, and mention recent work [7] that proves a similar result in higher dimensions.

We begin by demonstrating that for any distribution on $m$ points, a linear classifier can achieve an error smaller than $1/2 - \gamma$, where $\gamma = \Omega(1/m)$. In view of our comments above, such a fast convergence of $\gamma$ to zero may be useless for generalization bounds. We then use our construction to show that, under certain regularity conditions, a value of $\gamma$, and thus of $\theta$, which is independent of $m$ can be established for one-dimensional problems.

Let $\{x_1, \ldots, x_m\}$ be points in $\mathbb{R}^d$, and denote by $\{y_1, \ldots, y_m\}$ their binary labels, i.e., $y_i \in \{-1, +1\}$. A linear decision rule takes the form $\hat{y}(x) = \mathrm{sgn}(a \cdot x + b)$, where $\cdot$ is the standard inner product in $\mathbb{R}^d$. Let $P \in \Delta^m$ be a probability measure on the $m$ points. The weighted misclassification error for a classifier $\hat{y}$ is $P_e(a, b) = \sum_{i=1}^{m} P_i I(y_i \neq \hat{y}_i)$. For technical reasons, we prefer to use the expression $1 - 2P_e = \sum_{i=1}^{m} P_i y_i \hat{y}_i$. Obviously if $1 - 2P_e \gtrsim \epsilon$ we have that $P_e \lesssim \frac{1}{2} - \frac{\epsilon}{2}$.

**Lemma 1** *For any sample of $m$ distinct points, $S = \{(x_i, y_i)\}_{i=1}^{m} \in (\mathbb{R}^d \times \{-1, +1\})^m$, and a probability measure $P \in \Delta^m$ on $S$, there is some $a \in \mathbb{R}^d$ and $b \in \mathbb{R}$ such that the weighted misclassification error of the linear classifier $\hat{y} = \mathrm{sgn}(a \cdot x + b)$ is bounded away from $1/2$, in particular $\sum_{i=1}^{m} P_i I(y_i \neq \hat{y}_i) \leq \frac{1}{2} - \frac{1}{4m}$.*

**Proof** The basic idea of the proof is to project a finite number of points onto a line $h$ so that no two points coincide. Since there is at least one point $x$ whose weight is not smaller than $1/m$, we consider the four possible linear classifiers defined by $h$ with boundaries near $x$ (at both sides of it and with opposite sign), and show that one of these yields the desired result. We proceed to the detailed proof. Fix a probability vector $P = (P_1, \ldots, P_m) \in \Delta^m$. We may assume w.l.o.g that all the $x_i$ are different, or we can merge two elements and get $m - 1$ points. First, observe that if $|\sum_{i=1}^{m} P_i y_i| \geq \frac{1}{2m}$, then the problem is trivially solved. To see this, denote by $S_\pm$ the sub-samples of $S$ labelled by $\pm 1$ respectively. Assume, for example, that $\sum_{i \in S_+} P_i \geq \sum_{i \in S_-} P_i + \frac{1}{2m}$. Then the choice $a = 0, b = 1$, namely $\hat{y}_i = 1$ for all $i$, implies that $\sum_i P_i y_i \hat{y}_i \geq \frac{1}{2m}$. Similarly, the choice $a = 0, b = -1$ solves the problem if $\sum_{i \in S_-} P_i \geq \sum_{i \in S_+} P_i + \frac{1}{2m}$. Thus, we can assume, without loss of generality, that $|\sum_{i=1}^{m} P_i y_i| < \frac{1}{2m}$. Next, note that there exists a direction $u$ such that $i \neq j$ implies that $u \cdot x_i \neq u \cdot x_j$. This can be seen by the following argument. Construct all one-dimensional lines containing two data points or more; clearly the number of such lines is at most $m(m-1)/2$. It is then obvious that any line, which is *not* perpendicular to any of these lines obeys the required condition. Let $x_i$ be a data-point for which $P_i \geq 1/m$, and set $\epsilon$ to be a positive number such that $0 < \epsilon < \min\{|u \cdot x_i - u \cdot x_j| : i, j \in 1, \ldots, m\}$. Such an $\epsilon$ always exists since the points are assumed to be distinct. Note the following trivial algebraic fact:

$$|A + B| < \delta_1 \quad \& \quad A > \delta_2 \quad \Rightarrow \quad A - B > 2\delta_2 - \delta_1. \tag{3}$$

For each $j = 1, 2, \ldots, m$ let the classification be given by $\hat{y}_j = \mathrm{sgn}(u \cdot x_j + b)$, where the bias $b$ is given by $b = -u \cdot x_i + \epsilon y_i$. Then clearly $\hat{y}_i = y_i$ and $\hat{y}_j = \mathrm{sgn}(u \cdot x_j - u \cdot x_i)$, and therefore $\sum_j P_j y_j \hat{y}_j = P_i + \sum_{j \neq i} P_j y_j \mathrm{sgn}(u \cdot x_j - u \cdot x_i)$. Let $A = P_i$ and $B = \sum_{j \neq i} P_j y_j \mathrm{sgn}(u \cdot x_j - u \cdot x_i)$. If $|A + B| \geq 1/2m$ we are done. Otherwise, if $|A + B| < 1/2m$, consider the classifier $y'_j = \mathrm{sgn}(-u \cdot x_j + b')$, with $b' = u \cdot x_i + \epsilon y_i$ (note that $y'_i = y_i$ and $y'_j = -\hat{y}_j$, $j \neq i$). Using (3) with $\delta_1 = 1/2m$ and $\delta_2 = 1/m$ the claim follows. ∎

We comment that the upper bound in Lemma 1 may be improved to $1/2 - 1/(4(m-1))$, $m \geq 2$, using a more refined argument.

**Remark 1** Lemma 1 implies that an error of $1/2 - \gamma$, where $\gamma = \Omega(1/m)$, can be guaranteed for any set of arbitrarily weighted points. It is well known that the problem of finding a linear classifier with minimal classification error is NP-hard (in $d$) [5]. Moreover, even the problem of approximating the optimal solution is NP-hard [2]. Since the algorithm described in Lemma 1 is clearly polynomial (in $m$ and $d$), there seems to be a transition as a function of $\gamma$ between the class NP and P (assuming, as usual, that they are different). This issue warrants further investigation.

While the result given in Lemma 1 is interesting, its generality precludes its usefulness for bounding generalization error. This can be seen by observing that the theorem guarantees the given margin even in the case where the labels $y_i$ are drawn uniformly at random from $\{\pm 1\}$, in which case no generalization can be expected. In order to obtain a more useful result, we need to restrict the complexity of the data distribution. We do this by imposing constraints on the types of decision regions characterizing the data. In order to generate complex, yet tractable, decision regions we consider a multi-linear mapping from $\mathbb{R}^d$ to $\{-1, 1\}^k$, generated by the $k$ hyperplanes $\mathcal{P}_i = \{x : w_i x + w_{i0}, x \in \mathbb{R}^d\}, i = 1, \ldots, k$, as in the first hidden layer of a neural network. Such a mapping generates a partition of the input space $\mathbb{R}^d$ into $M$ *connected components*, $\{\mathbb{R}^d \setminus \cup_{i=1}^k \mathcal{P}_i\}$, each characterized by a unique binary vector of length $k$. Assume that the weight vectors $(w_i, w_{i0}) \in \mathbb{R}^{d+1}$ are in general position. The number of connected components is given by (e.g., Lemma 3.3. in [1]) $C(k, d+1) = 2\sum_{i=0}^{d} \binom{k-1}{i}$. This number can be bounded from below by $2\binom{k-1}{d}$, which in turn is bounded below by $2((k-1)/d)^d$. An upper bound is given by $2(e(k-1)/d)^d$, $m \geq d$. In other words, $C(k, d+1) = \Theta\left((k/d)^d\right)$. In order to generate a binary classification problem, we observe that there exists a binary function from $\{-1, 1\}^k \mapsto \{-1, 1\}$, characterized by these $M$ decision regions. This can be seen as follows. Choose an arbitrary connected component, and label it by $+1$ (say). Proceed by labelling all its neighbors by $-1$, where neighbors share a common boundary (a $(d-1)$-dimensional hyperplane in $d$ dimensions). Proceeding by induction, we generate a binary classification problem composed of exactly $M$ decision regions. Thus, we have constructed a binary classification problem, characterized by at least $2\binom{k-1}{d} \geq 2((k-1)/d))^d$ decision regions. Clearly as $k$ becomes arbitrarily large, very elaborate regions are formed.

We now apply these ideas, together with Lemma 1, to a one dimensional problem. Note that in this case the partition is composed of intervals.

**Theorem 1** *Let $\mathcal{F}$ be a class of functions from $\mathbb{R}$ to $\{\pm 1\}$ which partitions the real line into at most $k$ intervals, $k \geq 2$. Let $\mu$ be an arbitrary probability measure on $\mathbb{R}$. Then for any $f \in \mathcal{F}$ there exist $a, \tau^* \in \mathbb{R}$ for which,*

$$\mu\{x : f(x)\mathrm{sgn}(ax - \tau^*) = 1\} \geq \frac{1}{2} + \frac{1}{4k} \qquad (4)$$

**Proof** Let a function $f$ be given, and denote its connected components by $I_1, \ldots, I_k$, that is $I_1 = [-\infty, l_1), I_2 = [l_1, l_2), I_3 = [l_2, l_3)$, and so on until $I_k = [l_{k-1}, \infty]$, with $-\infty = l_0 < l_1 < l_2 < \cdots < l_{k-1}$. Associate with every interval a point in $\mathbb{R}$,

$$x_1 = l_1 - 1, x_2 = (l_1 + l_2)/2, \ldots, x_{k-1} = (l_{k-2} + l_{k-1})/2, x_k = l_{k-1} + 1,$$

a weight $\mu_i = \mu(I_i), i = 1, \ldots, k$, and a label $f(x_i) \in \{\pm 1\}$. We now apply Lemma 1 to conclude that there exist $a \in \{\pm 1\}$ and $\tau \in \mathbb{R}$ such that $\sum_{i=1}^k \mu_i f(x_i)\mathrm{sgn}(ax_i - \tau) > 1/(4k)$. The value of $\tau$ lies between $l_i$ and $l_{i+1}$ for some $i \in \{0, 1, \ldots, k-1\}$ (recall that $l_0 = -\infty$). We identify $\tau^*$ of (4) as $l_{i+1}$. This is the case since by choosing this $\tau^*$, $f(x)$ in any segment $I_i$ is equal to $f(x_i)$ so we have that $\mu\{x : f(x)\mathrm{sgn}(ax - \tau^*) = 1\} = \frac{1}{2} + \sum_{i=1}^k \mu_i f(x_i)\mathrm{sgn}(ax_i - \tau^*) \geq \frac{1}{2} + \frac{1}{4k}$. ∎

Note that the result in Theorem 1 is in fact more general than we need, as it applies to arbitrary distributions, rather than distributions over a finite set of points. An open problem at this point is whether a similar result applies to $d$-dimensional problems. We conjecture that in $d$ dimensions $\gamma$ behaves like $k^{-l(d)}$ for some function $l$, where $k$ is a measure for the number of homogeneous convex regions defined by the data (a homogeneous region is one in which all points possess identical labels).

While we do not have a general proof at this stage, we have recently shown [7] that the conjecture holds under certain natural conditions on the data. This result implies that, at least under appropriate conditions, boosting-like algorithm are expected to have excellent generalization performance. To provide some motivation, we present results of some numerical simulations for two-dimensional problems. For this simulation we used random lines to generate a partition of the unit square in $\mathbb{R}^2$. We then drew 1000 points at random from the unit square and assigned them labels according to the partition. Finally, in order to have a non-trivial problem we made sure that the cumulative weights of each class are equal. We then calculated the optimal linear classifier by exhaustive search. In Figure 1(b) we show a sample decision region with 93 regions. Figure 1(a) shows the dependence of $\gamma$ on the number of regions $k$. As it turns out there is a significant logarithmic dependence between $\gamma$ and $k$, which leads us to conjecture that $\gamma \sim C k^{-l} + E$ for some $C, l$ and $E$. In the presented case it turns out that $l = 3$ turns out to fit our model well. It is important to note, however, that the procedure described above only supports our claim in an average-case, rather than worst-case, setting as is needed.

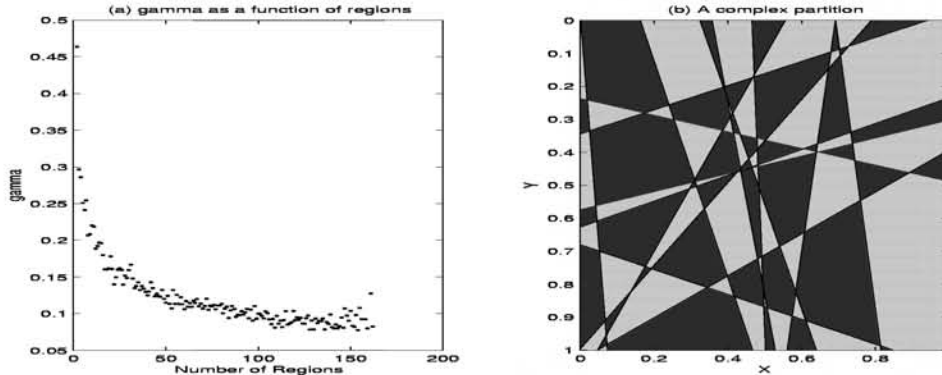

Figure 1: (a) $\gamma$ as a function of the number of regions. (b) A typical complex partition of the unit square used in the simulations.

## 3   Improved Convergence Rates

In Section 2 we proved that under certain conditions a weak learner exists with a sufficiently large margin, and thus the first term in (1) indeed converges to zero. We now analyze the second term in (1) and show that it may be made to converge considerably faster, *if* the first term is made somewhat larger. First, we briefly recall the framework introduced recently by Mason *et al.* [8]. These authors begin by introducing the notion of a B-admissible family of functions. For completeness we repeat their definition.

**Definition 1** (Definition 2 in [8]) *A family $\{C_N : N \in \mathbb{N}\}$ of margin cost functions is B-admissible for $B \geq 0$ if for all $N \in \mathbb{N}$ there is an interval $Y \subset \mathbb{R}$ of length no more than $B$ and a function $\Psi_N : [-1, 1] \mapsto Y$ that satisfies*

$$\mathrm{sgn}(-\alpha) \leq \mathbf{E}_{Z \sim Q_{N,\alpha}}[\Psi_N(Z)] \leq C_N(\alpha)$$

*for all $\alpha \in [-1, 1]$, where $\mathbf{E}_{Z \sim Q_{N,\alpha}}(\cdot)$ denotes the expectation when $Z$ is chosen randomly as $Z = (1/N) \sum_{i=1}^{N} Z_i$, and $\mathbf{P}(Z_i = 1) = (1 + \alpha)/2$.*

Denote the convex hull of a class $H$ by $\mathrm{co}(H)$. The main theoretical result in [8] is the following lemma.

**Lemma 2** ([8], Theorem 3) *For any $B-$admissible family $\{C_N : N \in \mathbb{N}\}$ of margin functions, for any binary hypothesis class of VC dimension $d_v$ and any distribution $D$ on $\mathcal{X} \times \{-1, +1\}$, with probability at least $1 - \delta$ over a random sample $S$ of $m$ examples drawn at random according to $D$, every $N$ and every $f \in \mathrm{co}(H)$ satisfies $\mathbf{P}_D[yf(x) \leq 0] \leq \mathbf{E}_S[C_N(yf(x)] + \epsilon_N$, where $\epsilon_N = O\left(\left[(B^2 N d_v \log m + \log(N/\delta))/m\right]^{1/2}\right)$.*

**Remark 2** The most appealing feature of Lemma 2, as of other results for convex combinations, is the fact that the bound does *not* depend on the number of hypotheses from $H$ defining $f \in \mathrm{co}(H)$, which may in fact be infinite. Using standard VC results (e.g. [11]) would lead to useless bounds, since the VC dimension of these classes is often huge (possibly infinite).

Lemma 2 considers binary hypotheses. Since recent works has demonstrated the effectiveness of using real valued hypotheses, we consider the case where the weak classifiers may be confidence-rated, i.e., taking values in $[-1, 1]$ rather than $\{\pm 1\}$. We first extend Lemma 2 to confidence-rated classifiers. Note that the variables $Z_i$ in Definition 1 are no longer binary in this case.

**Lemma 3** *Let the conditions of Lemma 2 hold, except that $H$ is a class of real valued functions from $\mathcal{X}$ to $[-1, +1]$ of pseudo-dimension $d_p$. Assume further that $\Psi_N$ in Definition 1 obeys a Lipschitz condition of the form $|\Psi_N(x) - \Psi_N(x')| \leq L|x - x'|$ for every $x, x' \in X$. Then with probability at least $1 - \delta$, $\mathbf{P}_D[yf(x) \leq 0] \leq \mathbf{E}_S[C_N(yf(x)] + \epsilon_N$, where $\epsilon_N = O\left(\left[(LB^2 N d_p \log m + \log(N/\delta))/m\right]^{1/2}\right)$.*

**Proof** The proof is very similar to the proof of Theorem 2, and will be omitted for the sake of brevity. ∎

It is well known that in the standard setting where $C_N$ is replaced by the empirical classification error, improved rates, replacing $O(\sqrt{\log m/m})$ by $O(\log m/m)$, are possible in two situations: (i) if the minimal value of $C_N$ is zero (the restricted model of [1]), and (ii) if the empirical error is replaced by $(1+\alpha)C_N$ for some $\alpha > 0$. The latter case is especially important in a model selection setup, where nested classes of hypothesis functions are considered, since in this case one expects that, with high probability, $C_N$ becomes smaller as the classes become more complex. In this situation, case (ii) provides better overall bounds, often leading to the optimal minimax rates for non parametric problems (see a discussion of these issues in Sec. 15.4 of [1]).

We now establish a faster convergence rate to a slightly larger value than $\mathbf{E}_S[C_N(Yf(X))]$. In situations where the latter quantity approaches zero, the overall convergence rate may be improved, as discussed above. We consider cost functions $C_N(\alpha)$, which obey the condition

$$C_N(\alpha) \leq (1 + \beta_N)I(\alpha < 0) + \eta_N \qquad (\beta_N > 0, \eta_N > 0). \qquad (5)$$

for some positive $\beta_N$ and $\eta_N$ (see [8] for details on legitimate cost functions).

**Theorem 2** *Let $D$ be a distribution over $\mathcal{X} \times \{-1, +1\}$, and let $S$ be a sample of $m$ points chosen independently at random according to $D$. Let $d_p$ be the pseudo-dimension of the class $H$, and assume that $C_N(\alpha)$ obeys condition (5). Then for sufficiently large $m$, with probability at least $1-\delta$, every function $f \in \mathrm{co}(H)$ satisfies the following bound for every $0 < \alpha < 1/\beta_N$*

$$\mathbf{P}_D[Yf(X) \leq 0] \leq \left(\frac{1+\alpha}{1-\alpha\beta_N}\right) \mathbf{E}_S[C_N(Yf(X))] + O\left(\frac{d_p N \log \frac{m}{d_p} + \log \frac{1}{\delta}}{m\alpha/(1+2\alpha)}\right).$$

**Proof** The proof combines two ideas. First, we use the method of [8] to transform the problem from co($H$) to a discrete approximation of it. Then, we use recent results for relative uniform deviations of averages from their means [3]. Due to lack of space, we defer the complete proof to the full version of the paper.

## 4 Discussion

In this paper we have presented two main results pertaining to the theory of boosting. First, we have shown that, under reasonable conditions, an effective weak classifier exists for one dimensional problems. We conjectured, and supported our claim by numerical simulations, that such a result holds for multi-dimensional problems as well. The non-trivial extension of the proof to multiple dimensions can be found in [7]. Second, using recent advances in the theory of uniform convergence and boosting we have presented bounds on the generalization error, which may, under certain conditions, be significantly better than standard bounds, being particularly useful in the context of model selection.

**Acknowledgment** We thank Shai Ben-David and Yoav Freund for helpful discussions.

## References

[1] M. Anthony and P.L. Bartlett. *Neural Network Learning: Theoretical Foundations*. Cambridge University Press, 1999.

[2] P. Bartlett and S. Ben-David. On the hardness of learning with neural networks. In *Proceedings of the fourth European Conference on computational Learning Theory*, 99.

[3] P. Bartlett and G. Lugosi. An inequality for uniform deviations of sample averages from their means. *Statistics and Probability Letters*, 44:55–62, 1999.

[4] T. Hastie J. Friedman and R. Tibshirani. Additive logistic regression: a statistical view of boosting. *The Annals of Statistics*, To appear, 2000.

[5] D.S. Johnson and F.P. Preparata. The densest hemisphere problem. *Theoretical Computer Science*, 6:93–107, 1978.

[6] S. Mallat and Z. Zhang. Matching pursuit with time-frequencey dictionaries. *IEEE Trans. Signal Processing*, 41(12):3397–3415, December 1993.

[7] S. Mannor and R. Meir. On the existence of weak learners and applications to boosting. Submitted to *Machine Learning*

[8] L. Mason, P. Bartlett and J. Baxter. Improved generalization through explicit optimization of margins. *Machine Learning*, 2000. To appear.

[9] L. Mason, P. Bartlett, J. Baxter and M. Frean. Functional gradient techniques for combining hypotheses. In B. Schölkopf A. Smola, P. Bartlett and D. Schuurmans, editors, *Advances in Large Margin Classifiers*. MIT Press, 2000.

[10] R.E. Schapire, Y. Freund, P. Bartlett and W.S. Lee. Boosting the margin: a new explanation for the effectiveness of voting methods. *The Annals of Statistics*, 26(5):1651–1686, 1998.

[11] V. N. Vapnik. *Estimation of Dependences Based on Empirical Data*. Springer Verlag, New York, 1982.
